# MICROELECTRONIC IMPLEMENTATIONS OF CONNECTIONIST NEURAL NETWORKS

Stuart Mackie, Hans P. Graf, Daniel B. Schwartz, and John S. Denker

AT&T Bell Labs, Holmdel, NJ 07733

## Abstract

In this paper we discuss why special purpose chips are needed for useful implementations of connectionist neural networks in such applications as pattern recognition and classification. Three chip designs are described: a hybrid digital/analog programmable connection matrix, an analog connection matrix with adjustable connection strengths, and a digital pipelined best-match chip. The common feature of the designs is the distribution of arithmetic processing power amongst the data storage to minimize data movement.

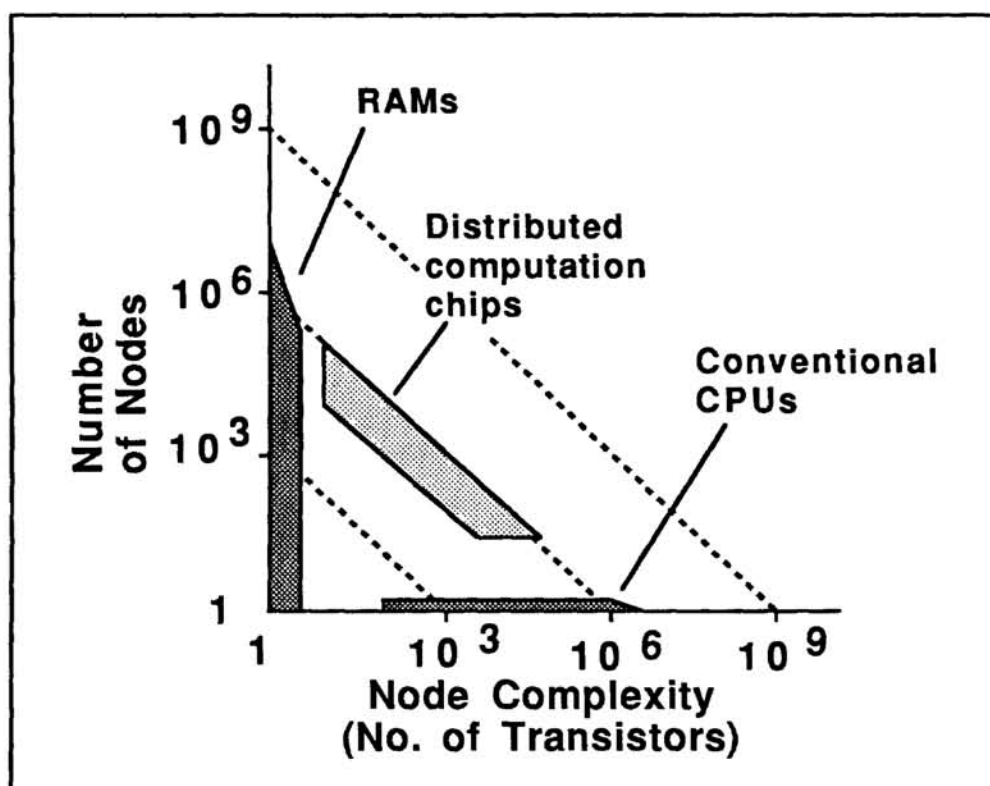

Figure 1. A schematic graph of addressable node complexity and size for conventional computer chips. Memories can contain millions of very simple nodes each with a very few transistors but with no processing power. CPU chips are essentially one very complex node. Neural network chips are in the distributed computation region where chips contain many simple fixed instruction processors local to data storage. (After Reece and Treleaven[1])

# Introduction

It is clear that conventional computers lag far behind organic computers when it comes to dealing with very large data rates in problems such as computer vision and speech recognition. Why is this? The reason is that the brain performs a huge number of operations in parallel whereas in a conventional computer there is a very fast processor that can perform a variety of instructions very quickly, but operates on only two pieces of data at a time.

The rest of the many megabytes of RAM is idle during any instruction cycle. The duty cycle of the processor is close to 100%, but that of the stored data is very close to zero. If we wish to make better use of the data, we have to distribute processing power amongst the stored data, in a similar fashion to the brain. Figure 1 illustrates where distributed computation chips lie in comparison to conventional computer chips as regard number and complexity of addressable nodes per chip.

In order for a distributed strategy to work, each processing element must be small in order to accommodate many on a chip, and communication must be local and hardwired. Whereas the processing element in a conventional computer may be able to execute many hundred different operations, in our scheme the processor is hard-wired to perform just one. This operation should be tailored to some particular application. In neural network and pattern recognition algorithms, the dot products of an input vector with a series of stored vectors (referred to as features or memories) is often required. The general calculation is:

$$\text{Sum of Products} \qquad V \cdot F(i) = \sum_j v_j f_{ij}$$

where $V$ is the input vector and $F(i)$ is one of the stored feature vectors. Two variations of this are of particular interest. In feature extraction, we wish to find all the features for which the dot product with the input vector is greater than some threshold $T$, in which case we say that such features are *present* in the input vector.

$$\text{Feature Extraction} \qquad V \cdot F(i) = \sum_j v_j f_{ij}$$

In pattern classification we wish to find the stored vector that has the largest dot product with the input vector, and we say that the the input is a *member of the class* represented by that feature, or simply that that stored vector is *closest* to input vector.

$$\text{Classification} \qquad \max(V \cdot F(i) = \sum_j v_j f_{ij}$$

The chips described here are each designed to perform one or more of the above functions with an input vector and a number of feature vectors in parallel. The overall strategy may be summed up as follows: we recognize that in typical pattern recognition applications, the feature vectors need to be changed infrequently compared to the input

vectors, and the calculation that is performed is fixed and low-precision, we therefore distribute simple fixed-instruction processors throughout the data storage area, thus minimizing the data movement and optimizing the use of silicon. Our ideal is to have every transistor on the chip doing something useful during every instruction cycle.

## Analog Sum-of-Products

Using an idea slightly reminiscent of synapses and neurons from the brain, in two of the chips we store elements of features as connections from input wires on which the elements of the input vectors appear as voltages to summing wires where a sum-of-products is performed. The voltage resulting from the current summing is applied to the input of an amplifier whose output is then read to determine the result of the calculation. A schematic arrangement is shown in Figure 2 with the vertical inputs connected to the horizontal summing wires through resistors chosen such that the conductance is proportional to the magnitude of the feature element. When both positive and negative values are required, inverted input lines are also necessary. Resistor matrices have been fabricated using amorphous silicon connections and metal linewidths. These were programmed during fabrication by electron beam lithography to store names using the distributed feedback method described by Hopfield[2,3]. This work is described more fully elsewhere.[4,5] Hard-wired resistor matrices are very compact, but also very inflexible. In many applications it is desirable to be able to reprogram the matrix without having to fabricate a new chip. For this reason, a series of programmable chips has been designed.

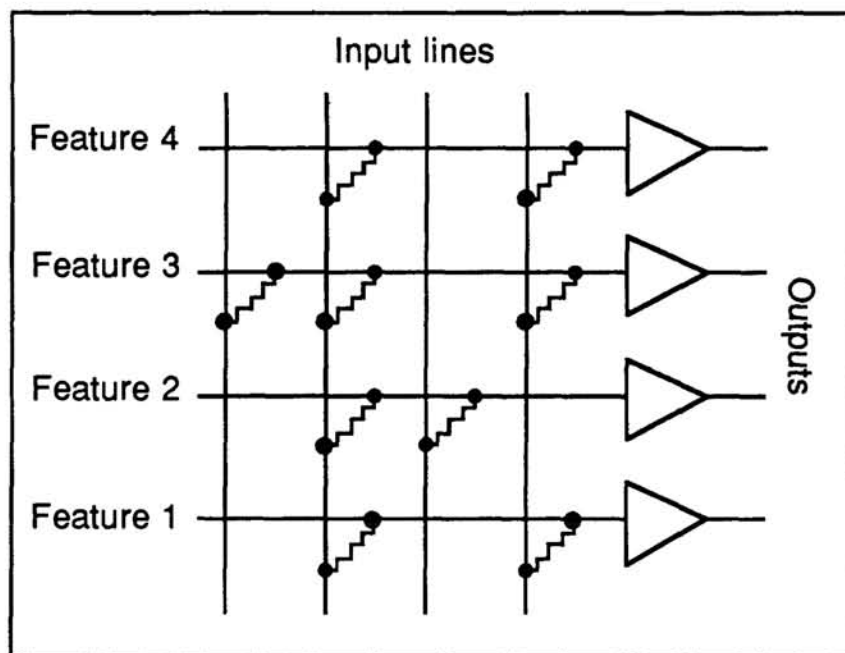

Figure 2. A schematic arrangement for calculating parallel sum-of-products with a resistor matrix. Features are stored as connections along summing wires and the input elements are applied as voltages on the input wires. The voltage generated by the current summing is thresholded by the amplifier whose output is read out at the end of the calculation. Feedback connections may be

made to give mutual inhibition and allow only one feature amplifier to turn on, or allow the matrix to be used as a distributed feedback memory.

## Programmable Connection Matrix

Figure 3 is a schematic diagram of a programmable connection using the contents of two RAM cells to control current sinking or sourcing into the summing wire. The switches are pass transistors and the 'resistors' are transistors with gates connected to their drains. Current is sourced or sunk if the appropriate RAM cell contains a '1' and the input $V_i$ is high thus closing both switches in the path. Feature elements can therefore take on values (a,0,-b) where the values of a and b are determined by the conductivities of the n- and p-transistors obtained during processing. A matrix with 2916 such connections allowing full interconnection of the inputs and outputs of 54 amplifiers was designed and fabricated in 2.5µm CMOS (Figure 4). Each connection is about 100×100µm, the chip is 7×7mm and contains about 75,000 transistors. When loaded with 49 49-bit features (7×7 kernel), and presented with a 49-bit input vector, the chip performs 49 dot products in parallel in under 1µs. This is equivalent to 2.4 billion bit operations/sec. The flexibility of the design allows the chip to be operated in several modes. The chip was programmed as a distributed feedback memory (associative memory), but this did not work well because the current sinking capability of the n-type transistors was 6 times that of the p-types. An associative memory was implemented by using a 'grandmother cell' representation, where the memories were stored along the input lines of amplifiers, as for feature extraction, but mutually inhibitory connections were also made that allowed only one output to turn on. With 10 stored vectors each 40 bits long, the best match was found in 50-600ns, depending on the data. The circuit can also be programmed to recognize sequences of vectors and to do error correction when vectors were omitted or wrong vectors were inserted into the sequences. The details of operation of the chip are described more fully elsewhere[6]. This chip has been interfaced to a UNIX minicomputer and is in everyday use as an accelerator for feature extraction in optical character recognition of hand-written numerals. The chip speeds up this time consuming calculation by a factor of more than 1000. The use of the chip enables experiments to be done which would be too time consuming to simulate.

Experience with this device has led to the design of four new chips, which are currently being tested. These have no feedback capability and are intended exclusively for feature extraction. The designs each incorporate new features which are being tested separately, but all are based on a connection matrix which stores 46 vectors each 96 bits long. The chip will perform a full parallel calculation in 100ns.

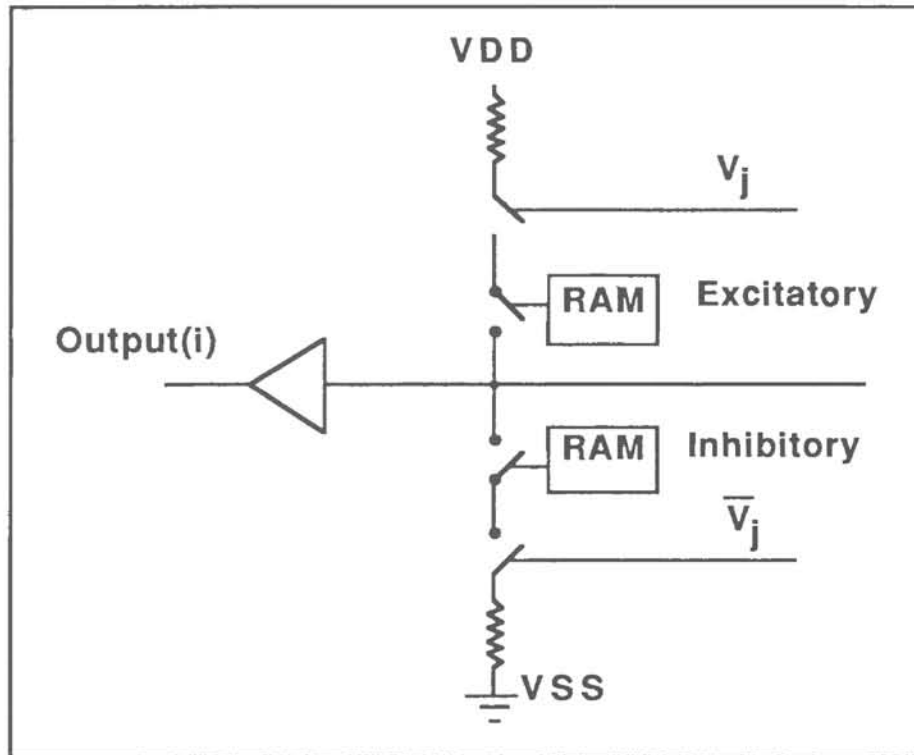

Figure 3. Schematic diagram of a programmable connection. A current sourcing or sinking connection is made if a RAM cell contains a '1' and the input $V_i$ is high. The currents are summed on the input wire of the amplifier.

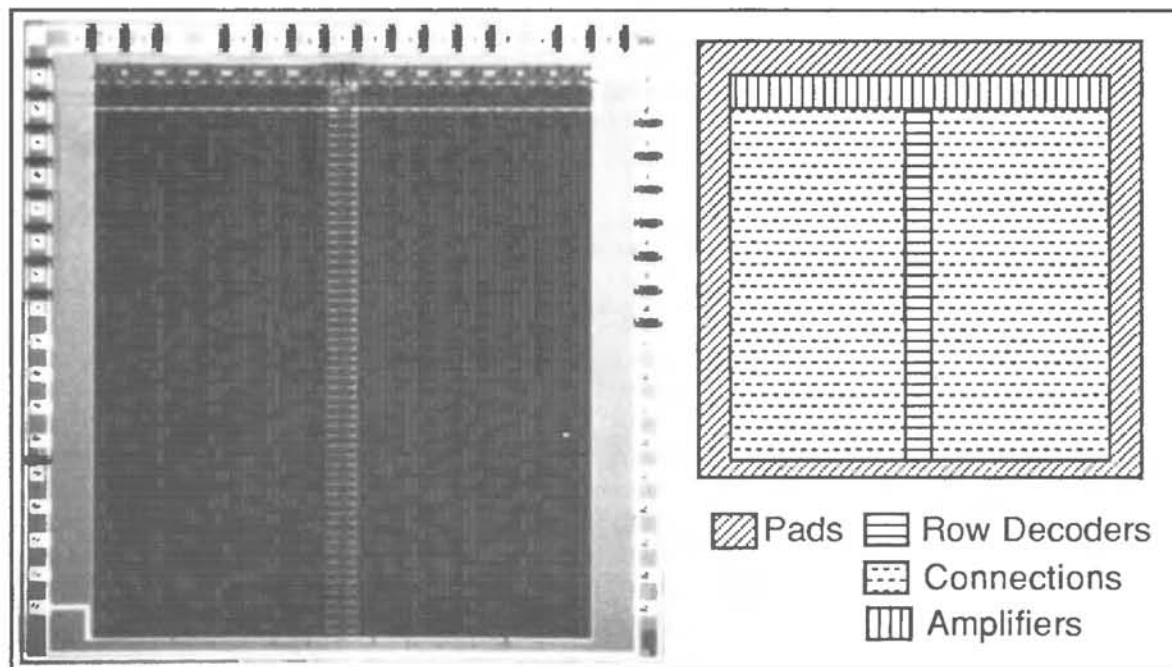

Figure 4. Programmable connection matrix chip. The chip contains 75,000 transistors in 7x7mm, and was fabricated using 2.5μm design rules.

## Adaptive Connection Matrix

Many problems require analog depth in the connection strengths, and this is especially important if the chip is to be used for learning, where small adjustments are required during training. Typical approaches which use transistors sized in powers of two to give conductance variability take up an area equivalent to the same number of minimum sized transistors as the dynamic range, which is expensive in area and enables only a few connections to be put on a chip. We have designed a fully analog connection based on a DRAM structure that can be fabricated using conventional CMOS technology. A schematic of a connection and a connection matrix is shown in Figure 5. The connection strength is represented by the difference in voltages stored on two MOS capacitors. The capacitors are 33μm on edge and lose about 1% of their charge in five minutes at room temperature. The leakage rate can be reduced by three orders of magnitude by cooling the the capacitors to –50°C and by five orders of magnitude by cooling to –100°C. The output is a current proportional to the product of the input voltage and the connection strength. The output currents are summed on a wire and are sent off chip to external amplifiers. The connection strengths can be adjusted using transferring charge between the capacitors through a chain of transistors. The connections strengths may be of either polarity and it is expected that the connections will have about 7 bits of analog depth. A chip has been designed in 1.25μm CMOS containing 1104 connections in an array with 46 inputs and 24 outputs.

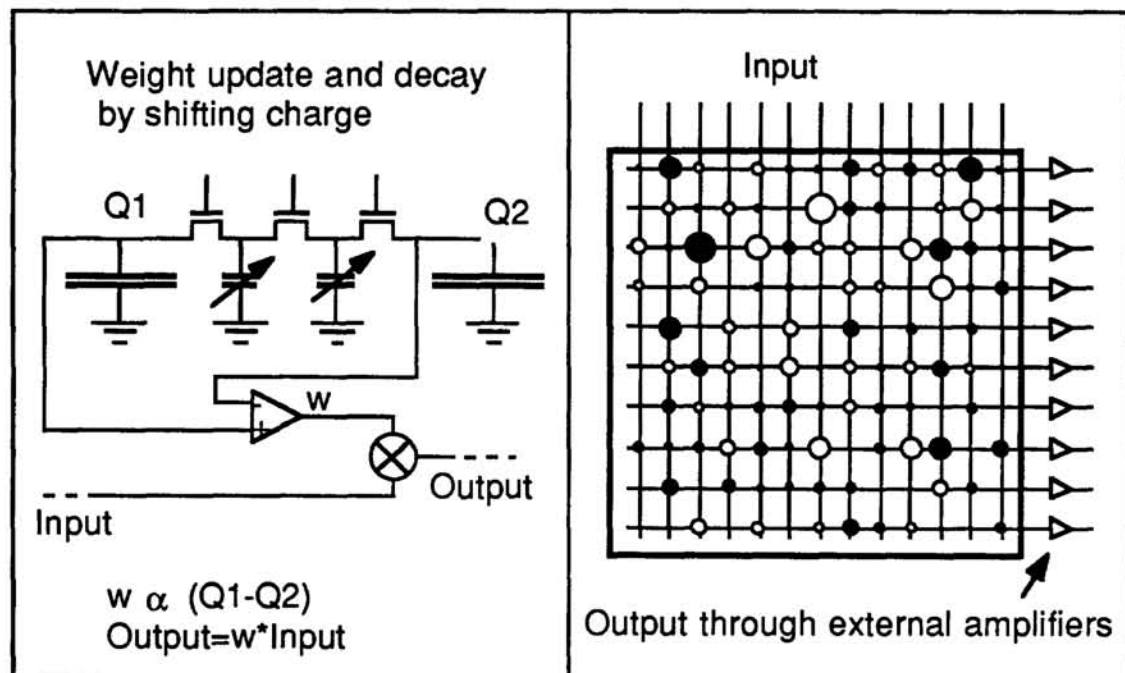

Figure 5. Analog connection. The connection strength is represented by the difference in voltages stored on two capacitors. The output is a current proprtional to the product of the input voltage and the connection strength.

Each connection is 70×240μm. The design has been sent to foundry, and testing is expected to start in April 1988. The chip has been designed to perform a network calculation in <30ns, i.e., the chip will perform at a rate of 33 billion multiplies/sec. It can be used simply as a fast analog convolver for feature extraction, or as a learning

engine in a gradient descent algorithm using external logic for connection strength adjustment. Because the inputs and outputs are true analog, larger networks may be formed by tiling chips, and layered networks may be made by cascading through amplifiers acting as hidden units.

## Digital Classifier Chip

The third design is a digital implementation of a classifier whose architecture is not a connectionist matrix. It is nearing completion of the design stage, and will be fabricated using 1.25μm CMOS. It calculates the largest five V·F(i) using an all-digital pipeline of identical processors, each attached to one stored word. Each processor is also internally pipelined to the extent that no stage contains more than two gate delays. This is important, since the throughput of the processor is limited by the speed of the slowest stage. Each processor calculates the Hamming distance (number of difference bits) between an input word and its stored word, and then compares that distance with each of the smallest 5 values previously found for that input word. An updated list of 5 best matches is then passed to the next processor in the pipeline. At the end of the pipeline the best 5 matches overall are output.

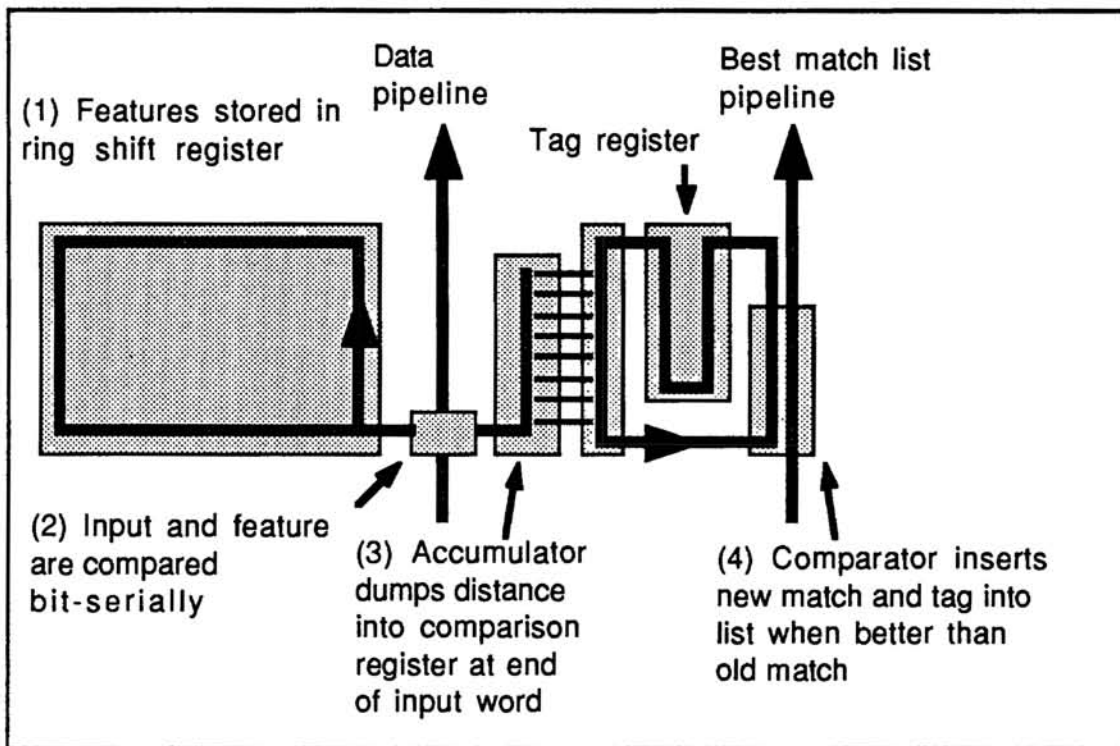

Fig. 6    Schematic of one of the 50 processors in the digital classifier chip. The Hamming distance of the input vector to the feature vector is calculated, and if better than one of the five best matches found so far, is inserted into the match list together with the tag and passed onto the next processor. At the end of the pipeline the best five matches overall are output.

The data paths on chip are one bit wide and all calculations are bit serial. This means that the processing elements and the data paths are compact and maximizes the number of stored words per chip. The layout of a single processor is shown in Fig. 6. The features are stored as 128-bit words in 8 16-bit ring shift registers and associated with each feature is a 14-bit tag or name string that is stored in a static register. The input vector passes through the chip and is compared bit-by-bit to each stored vector, whose shift registers are cycled in turn. The total number of bits difference is summed in an accumulator. After a vector has passed through a processor, the total Hamming distance is loaded into the comparison register together with the tag. At this time, the match list for the input vector arrives at the comparator. It is an ordered list of the 5 lowest Hamming distances found in the pipeline so far, together with associated tag strings. The distance just calculated is compared bit-serially with each of the values in the list in turn. If the current distance is smaller than one of the ones in the list, the output streams of the comparator are switched, having the effect of inserting the current match and tag into the list and deleting the previous fifth best match. After the last processor in the pipeline, the list stream contains the best five distances overall, together with the tags of the stored vectors that generated them. The data stream and the list stream are loaded into 16-bit wide registers ready for output. The design enables chips to be connected together to extend the pipeline if more than 50 stored vectors are required. The throughput is constant, irrespective of the number of chips connected together; only the latency increases as the number of chips increases.

The chip has been designed to operate with an on-chip clock frequency of at least 100MHz. This high speed is possible because stage sizes are very small and data paths have been kept short. The computational efficiency is not as high as in the analog chips because each processor only deals with one bit of stored data at a time. However, the overall throughput is high because of the high clock speed. Assuming a clock frequency of 100MHz, the chip will produce a list of 5 best distances with tag strings every 1.3µs, with a latency of about 2.5µs. Even if a thousand chips containing 50,000 stored vectors were pipelined together, the latency would be 2.5ms, low enough for most real time applications. The chip is expected to perform 5 billion bit operation/sec.

While it is important to have high clock frequencies on the chip, it is also important to have them much lower off the chip, since frequencies above 50MHz are hard to deal on circuit boards. The 16-bit wide communication paths onto and off the chip ensure that this is not a problem here.

## Conclusion

The two approaches discussed here, analog and digital, represent opposites in computational approach. In one, a single global computation is performed for each match, in the other many local calculations are done. Both the approaches have their advantages and it remains to be seen which type of circuit will be more efficient in applications, and how closely an electronic implementation of a neural network should resemble the highly interconnected nature of a biological network.

These designs represent some of the first distributed computation chips. They are characterized by having simple processors distributed amongst data storage. The operation performed by the processor is tailored to the application. It is interesting to note some of the reasons why these designs can now be made: minimum linewidths on

circuits are now small enough that enough processors can be put on one chip to make these designs of a useful size, sophisticated design tools are now available that enable a single person to design and simulate a complete circuit in a matter of months, and fabrication costs are low enough that highly speculative circuits can be made without requiring future volume production to offset prototype costs.

We expect a flurry of similar designs in the coming years, with circuits becoming more and more optimized for particular applications. However, it should be noted that the impressive speed gain achieved by putting an algorithm into custom silicon can only be done once. Further gains in speed will be closely tied to mainstream technological advances in such areas as transistor size reduction and wafer-scale integration. It remains to be seen what influence these kinds of custom circuits will have in useful technology since at present their functions cannot even be simulated in reasonable time. What can be achieved with these circuits is very limited when compared with a three dimensional, highly complex biological system, but is a vast improvement over conventional computer architectures.

The authors gratefully acknowledge the contributions made by L.D. Jackel, and R.E. Howard

# References

1 M. Reece and P.C. Treleaven, "Parallel Architectures for Neural Computers", Neural Computers, R. Eckmiller and C. v.d. Malsburg, eds (Springer-Verlag, Heidelberg, 1988)

2 J.J. Hopfield, Proc. Nat. Acad. Sci. **79**, 2554 (1982).

3 J.S. Denker, Physica **22D**, 216 (1986).

4 R.E. Howard, D.B. Schwartz, J.S. Denker, R.W. Epworth, H.P. Graf, W.E. Hubbard, L.D. Jackel, B.L. Straughn, and D.M. Tennant, IEEE Trans. Electron Devices **ED-34**, 1553, (1987)

5 H.P. Graf and P. deVegvar, "A CMOS Implementation of a Neural Network Model", in "Advanced Research in VLSI", Proceedings of the 1987 Stanford Conference, P. Losleben (ed.), (MIT Press 1987).

6 H.P. Graf and P. deVegvar, "A CMOS Associative Memory Chip Based on Neural Networks", Tech. Digest, 1987 IEEE International Solid-State Circuits Conference.
